# Recurrent Neural Networks for Missing or Asynchronous Data

**Yoshua Bengio** [*]
Dept. Informatique et
Recherche Opérationnelle
Université de Montréal
Montreal, Qc H3C-3J7
bengioy@iro.umontreal.ca

**Francois Gingras**
Dept. Informatique et
Recherche Opérationnelle
Université de Montréal
Montreal, Qc H3C-3J7
gingras@iro.umontreal.ca

## Abstract

In this paper we propose recurrent neural networks with feedback into the input units for handling two types of data analysis problems. On the one hand, this scheme can be used for static data when some of the input variables are missing. On the other hand, it can also be used for sequential data, when some of the input variables are missing or are available at different frequencies. Unlike in the case of probabilistic models (e.g. Gaussian) of the missing variables, the network does not attempt to model the distribution of the missing variables given the observed variables. Instead it is a more "discriminant" approach that fills in the missing variables for the sole purpose of minimizing a learning criterion (e.g., to minimize an output error).

## 1 Introduction

Learning from examples implies discovering certain relations between variables of interest. The most general form of learning requires to essentially capture the joint distribution between these variables. However, for many specific problems, we are only interested in predicting the value of certain variables when the others (or some of the others) are given. A distinction is therefore made between input variables and output variables. Such a task requires less information (and less parameters, in the case of a parameterized model) than that of estimating the full joint distribution. For example in the case of classification problems, a traditional statistical approach is based on estimating the conditional distribution of the inputs for each class, as well as the class prior probabilities (thus yielding the full joint distribution of inputs and classes). A more discriminant approach concentrates on estimating the class boundaries (and therefore requires less parameters), as for example with a feedforward neural network trained to estimate the output class probabilities given the observed variables.

However, for many learning problems, only some of the input variables are given for each particular training case, and the missing variables differ from case to case. The simplest way to deal with this missing data problem consists in replacing the missing values by their *unconditional mean*. It can be used with "discriminant" training algorithms such as those used with feedforward neural networks. However, in some problems, one can obtain better results by taking advantage of the dependencies between the input variables. A simple idea therefore consists

---

[*] also, AT&T Bell Labs, Holmdel, NJ 07733

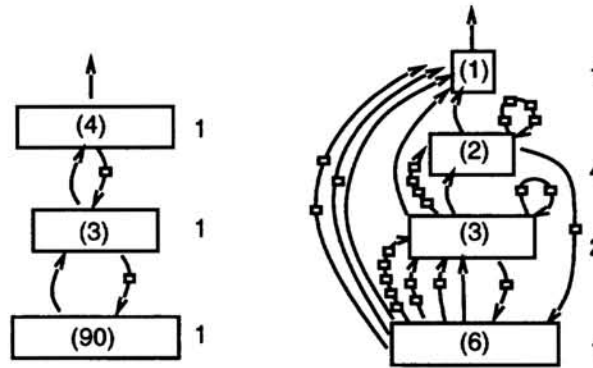

Figure 1: Architectures of the recurrent networks in the experiments. On the left a 90-3-4 architecture for static data with missing values, on the right a 6-3-2-1 architecture with multiple time-scales for asynchronous sequential data. Small squares represent a unit delay. The number of units in each layer is inside the rectangles. The time scale at which each layer operates is on the right of each rectangle.

in replacing the missing input variables by their *conditional* expected value, when the observed input variables are given. An even better scheme is to compute the expected output given the observed inputs, e.g. with a mixture of Gaussian. Unfortunately, this amounts to estimating the full joint distribution of all the variables. For example, with $n_i$ inputs, capturing the possible effect of each observed variable on each missing variable would require $O(n_i^2)$ parameters (at least one parameter to capture some co-occurrence statistic on each pair of input variables). Many related approaches have been proposed to deal with missing inputs using a Gaussian (or Gaussian mixture) model (Ahmad and Tresp, 1993; Tresp, Ahmad and Neuneier, 1994; Ghahramani and Jordan, 1994). In the experiments presented here, the proposed recurrent network is compared with a Gaussian mixture model trained with EM to handle missing values (Ghahramani and Jordan, 1994).

The approach proposed in section 2 is more economical than the traditional Gaussian-based approaches for two reasons. Firstly, we take advantage of hidden units in a recurrent network, which might be less numerous than the inputs. The number of parameters depends on the product of the number of hidden units and the number of inputs. The hidden units only need to capture the dependencies between input variables which have some dependencies, *and* which are useful to reducing the output error. The second advantage is indeed that training is based on optimizing the desired criterion (e.g., reducing an output error), rather than predicting as well as possible the values of the missing inputs. The recurrent network is allowed to relax for a few iterations (typically as few as 4 or 5) in order to fill-in some values for the missing inputs and produce an output. In section 3 we present experimental results with this approach, comparing the results with those obtained with a feedforward network.

In section 4 we propose an extension of this scheme to sequential data. In this case, the network is not relaxing: inputs keep changing with time and the network maps an input sequence (with possibly missing values) to an output sequence. The main advantage of this extension is that it allows to deal with sequential data in which the variables occur at different frequencies. This type of problem is frequent for example with economic or financial data. An experiment with asynchronous data is presented in section 5.

## 2  Relaxing Recurrent Network for Missing Inputs

Networks with feedback such as those proposed in (Almeida, 1987; Pineda, 1989) can be applied to learning a static input/output mapping when some of the inputs are missing. In both cases, however, one has to wait for the network to relax either to a fixed point (assuming it does find one) or to a "stable distribution" (in the case of the Boltzmann machine). In the case of fixed-point recurrent networks, the training algorithm assumes that a fixed point has been reached. The gradient with respect to the weights is then computed in order to move the fixed point to a more desirable position. The approach we have preferred here avoids such an assumption.

Instead it uses a more explicit optimization of the whole behavior of the network as it unfolds in time, fills-in the missing inputs and produces an output. The network is trained to minimize some function of its output by back-propagation through time.

---

**Computation of Outputs Given Observed Inputs**
**Given:**  input vector $u = [u_1, u_2, \ldots, u_{n_i}]$
**Result:**  output vector $y = [y_1, y_2, \ldots, y_{n_o}]$

    1. **Initialize for** $t = 0$:
      For $i = 1 \ldots n_u, x_{0,i} \leftarrow 0$
      For $i = 1 \ldots n_i$, if $u_i$ is missing then $x_{0,I(i)} \leftarrow E(i)$,
               Else $x_{0,I(i)} \leftarrow u_i$.
    2. **Loop over time:**
      For $t = 1$ to $T$
        For $i = 1 \ldots n_u$
          If $i = I(k)$ is an input unit and $u_k$ is not missing then
            $x_{t,i} \leftarrow u_k$
         Else
            $x_{t,i} \leftarrow (1 - \gamma)x_{t-1,i} + \gamma f(\sum_{l \in S_i} w_l x_{t-d_l, p_l})$
            where $S_i$ is a set of links from unit $p_l$ to unit $i$,
            each with weight $w_l$ and a discrete delay $d_l$
            (but terms for which $t - d_l < 0$ were not considered).
    3. **Collect outputs by averaging at the end of the sequence:**
      $y_i \leftarrow \sum_{t=1}^{T} v_t \, x_{t,O(i)}$

**Back-Propagation**
The back-propagation computation requires an extra set of variables $\dot{x}_t$ and $\dot{w}$, which will contain respectively $\frac{\partial C}{\partial x_t}$ and $\frac{\partial C}{\partial w}$ after this computation.
**Given:**  output gradient vector $\frac{\partial C}{\partial y}$
**Result:**  input gradient $\frac{\partial C}{\partial u}$ and parameter gradient $\frac{\partial C}{\partial w}$.

    1. **Initialize unit gradients using outside gradient:**
    Initialize $\dot{x}_{t,i} = 0$ for all $t$ and $i$.
    For $i = 1 \ldots n_o$, initialize $\dot{x}_{t,O(i)} \leftarrow v_t \frac{\partial C}{\partial y_i}$

    2. **Backward loop over time:**
    For $t = T$ to $1$
        For $i = n_u \ldots 1$
          If $i = I(k)$ is an input unit and $u_k$ is not missing then
            no backward propagation
        Else
            For $l \in S_i$
            If $t - d_l > 0$
              $\dot{x}_{t-d_l, p_l} \leftarrow \dot{x}_{t-d_l, p_l} + (1 - \gamma)\dot{x}_{t-d_l+1}$
                    $+ \gamma w_l \dot{x}_{t,i} f'(\sum_{l \in S_i} w_l x_{t-d_l, p_l})$
              $\dot{w}_l \leftarrow \dot{w}_l + \gamma f'(\sum_{l \in S_i} w_l x_{t-d_l, p_l}) x_{t-d_l, p_l}$
    3. **Collect input gradients:**
    For $i = 1 \ldots n_i$,
      If $u_i$ is missing, then
        $\frac{\partial C}{\partial u_i} \leftarrow 0$
      Else
        $\frac{\partial C}{\partial u_i} \leftarrow \sum_t \dot{x}_{t,I(i)}$

---

The observed inputs are clamped for the whole duration of the sequence. The missing units corresponding to missing inputs are initialized to their unconditional expectation and their value is then updated using the feedback links for the rest of the sequence (just as if they were hidden units). To help stability of the network and prevent it from finding periodic solutions (in which the outputs have a correct output only periodically), output supervision is given for several time steps. A fixed vector $v$, with $v_t > 0$ and $\sum_t v_t = 1$ specifies a weighing scheme that distributes

the responsibility for producing the correct output among different time steps. Its purpose is to encourage the network to develop stable dynamics which gradually converge toward the correct output (thus the weights $v_t$ were chosen to gradually increase with $t$).

The neuron transfer function was a hyperbolic tangent in our experiments. The inertial term weighted by $\gamma$ (in step 3 of the forward propagation algorithm below) was used to help the network find stable solutions. The parameter $\gamma$ was fixed by hand. In the experiments described below, a value of 0.7 was used, but near values yielded similar results.

This module can therefore be combined within a hybrid system composed of several modules by propagating gradient through the combined system (as in (Bottou and Gallinari, 1991)). For example, as in Figure 2, there might be another module taking as input the recurrent network's output. In this case the recurrent network can be seen as a feature extractor that accepts data with missing values in input and computes a set of features that are never missing. In another example of hybrid system the non-missing values in input of the recurrent network are computed by another, upstream module (such as the preprocessing normalization used in our experiments), and the recurrent network would provide gradients to this upstream module (for example to better tune its normalization parameters).

## 3   Experiments with Static Data

A network with three layers (inputs, hidden, outputs) was trained to classify data with missing values from the *audiology* database. This database was made public thanks to Jergen and Quinlan, was used by (Bareiss and Porter, 1987), and was obtained from the UCI Repository of machine learning databases (`ftp.ics.uci.edu:pub/machine-learning-databases`). The original database has 226 patterns, with 69 attributes, and 24 classes. Unfortunately, most of the classes have only 1 exemplar. Hence we decided to cluster the classes into four groups. To do so, the average pattern for each of the 24 classes was computed, and the K-Means clustering algorithm was then applied on those 24 prototypical class "patterns", to yield the 4 "superclasses" used in our experiments. The multi-valued input symbolic attributes (with more than 2 possible values) where coded with a "one-out-of-$n$" scheme, using $n$ inputs (all zeros except the one corresponding to the attribute value). Note that a missing value was represented with a special numeric value recognized by the neural network module. The inputs which were constant over the training set were then removed. The remaining 90 inputs were finally standardized (by computing mean and standard deviation) and transformed by a saturating non-linearity (a scaled hyperbolic tangent). The output class is coded with a "one-out-of-4" scheme, and the recognized class is the one for which the corresponding output has the largest value.

The architecture of the network is depicted in Figure 1 (left). The length of each relaxing sequence in the experiments was 5. Higher values would not bring any measurable improvements, whereas for shorter sequences performance would degrade. The number of hidden units was varied, with the best generalization performance obtained using 3 hidden units.

The recurrent network was compared with feedforward networks as well as with a mixture of Gaussians. For the feedforward networks, the missing input values were replaced by their unconditional expected value. They were trained to minimize the same criterion as the recurrent networks, i.e., the sum of squared differences between network output and desired output. Several feedforward neural networks with varying numbers of hidden units were trained. The best generalization was obtained with 15 hidden units. Experiments were also performed with no hidden units and two hidden layers (see Table 1). We found that the recurrent network not only generalized better but also learned much faster (although each pattern required 5 times more work because of the relaxation), as depicted in Figure 3.

The recurrent network was also compared with an approach based on a Gaussian and Gaussian mixture model of the data. We used the algorithm described in (Ghahramani and Jordan, 1994) for supervised leaning from incomplete data with the EM algorithm. The whole joint input/output distribution is modeled using a mixture model with Gaussians (for the inputs) and multinomial (outputs) components:

$$P(\mathbf{X} = \mathbf{x}, C = c) = \sum_j P(\omega_j) \frac{\mu_{jd}}{(2\pi)^{n/2}|\Sigma_j|^{1/2}} \exp\{-\frac{1}{2}(\mathbf{x} - \mu_j)'\Sigma_j^{-1}(\mathbf{x} - \mu_j)\}$$

where $\mathbf{x}$ is the input vector, $c$ the output class, and $P(\omega_j)$ the prior probability of component $j$ of the mixture. The $\mu_{jd}$ are the multinomial parameters; $\mu_j$ and $\Sigma_j$ are the Gaussian mean vector

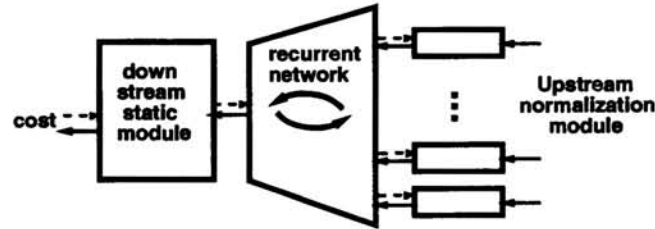

Figure 2: Example of hybrid modular system, using the recurrent network (middle) to extract features from patterns which may have missing values. It can be combined with upstream modules (e.g., a normalizing preprocessor, right) and downstream modules (e.g., a static classifier, left). Dotted arrows show the backward flow of gradients.

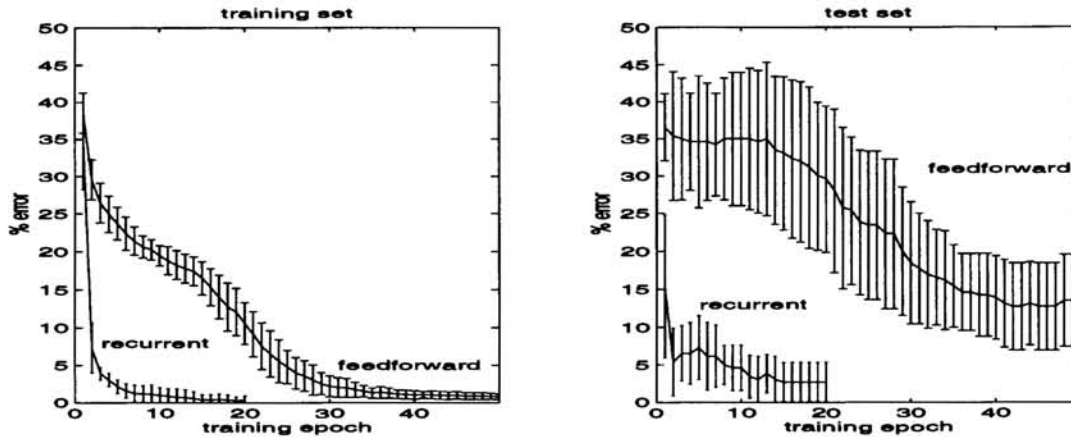

Figure 3: Evolution of training and test error for the recurrent network and for the best of the feedforward networks (90-15-4): average classification error w.r.t. training epoch, (with 1 standard deviation error bars, computed over 10 trials).

and covariance matrix for component $j$. Maximum likelihood training is applied as explained in (Ghahramani and Jordan, 1994), taking missing values into account (as additional missing variables of the EM algorithm).

For each architecture in Table 1, 10 training trials were run with a different subset of 200 training and 26 test patterns (and different initial weights for the neural networks). The recurrent network was clearly superior to the other architectures, probably for the reasons discussed in the conclusion. In addition, we have shown graphically the rate of convergence during training of the best feedforward network (90-15-4) as well as the best recurrent network (90-3-4), in Figure 3. Clearly, the recurrent network not only performs better at the end of training but also learns much faster.

## 4  Recurrent Network for Asynchronous Sequential Data

An important problem with many sequential data analysis problems such as those encountered in financial data sets is that different variables are known at different frequencies, at different times (phase), or are sometimes missing. For example, some variables are given daily, weekly, monthly, quarterly, or yearly. Furthermore, some variables may not even be given for some of the periods or the precise timing may change (for example the date at which a company reports financial performance my vary).

Therefore, we propose to extend the algorithm presented above for static data with missing values to the general case of sequential data with missing values or asynchronous variables. For time steps at which a low-frequency variable is not given, a missing value is assumed in input. Again, the feedback links from the hidden and output units to the input units allow the network

Table 1: Comparative performances of recurrent network, feedforward network, and Gaussian mixture density model on audiology data. The average percentage of classification error is shown after training, for both training and test sets, and the standard deviation in parenthesis, for 10 trials.

|  | Training set error | Test set error |
|---|---|---|
| 90-3-4 Recurrent net | 0.3(0.6) | 2.7(2.6) |
| 90-6-4 Recurrent net | 0(0) | 3.8(4) |
| 90-25-4 Feedforward net | 0.5(1.6) | 15(7.3) |
| 90-15-4 Feedforward net | 0.8(0.4) | 13.8(7) |
| 90-10-6-4 Feedforward net | 1(0.9) | 16(5.3) |
| 90-6-4 Feedforward net | 6(4.9) | 29(8.9) |
| 90-2-4 Feedforward net | 18.5(1) | 27(10) |
| 90-4 Feedforward net | 22(1) | 33(8) |
| 1 Gaussian | 35(1.6) | 38(9.3) |
| 4 Gaussians Mixture | 36(1.5) | 38(9.2) |
| 8 Gaussians Mixture | 36(2.1) | 38(9.3) |

to "complete" the missing data. The main differences with the static case are that the inputs and outputs vary with $t$ (we use $u_t$ and $y_t$ at each time step instead of $u$ and $y$). The training algorithm is otherwise the same.

## 5   Experiments with Asynchronous Data

To evaluate the algorithm, we have used a recurrent network with random weights, and feedback links on the input units to generate artificial data. The generating network has 6 inputs, 3 hidden and 1 outputs. The hidden layer is connected to the input layer (1 delay). The hidden layer receives inputs with delays 0 and 1 from the input layer and with delay 1 from itself. The output layer receives inputs from the hidden layer. At the initial time step as well as at 5% of the time steps (chosen randomly), the input units were clamped with random values to introduce some further variability. The missing values were then completed by the recurrent network. To generate asynchronous data, half of the inputs were then hidden with missing values 4 out of every 5 time steps. 100 training sequences and 50 test sequences were generated. The learning problem is therefore a sequence regression problem with missing and asynchronous input variables.

Preliminary comparative experiments show a clear advantage to completing the missing values (due to the the different frequencies of the input variables) with the recurrent network, as shown in Figure 4. The recognition recurrent network is shown on the right of Figure 1. It has multiple time scales (implemented with subsampling and oversampling, as in TDNNs (Lang, Waibel and Hinton, 1990) and reverse-TDNNs (Simard and LeCun, 1992)), to facilitate the learning of such asynchronous data. The static network is a time-delay neural network with 6 input, 8 hidden, and 1 output unit, and connections with delays 0, 2, and 4 from the input to hidden and hidden to output units. The "missing values" for slow-varying variables were replaced by the last observed value in the sequence. Experiments with 4 and 16 hidden units yielded similar results.

## 6   Conclusion

When there are dependencies between input variables, and the output prediction can be improved by taking them into account, we have seen that a recurrent network with input feedback can perform significantly better than a simpler approach that replaces missing values by their unconditional expectation. According to us, this explains the significant improvement brought by using the recurrent network instead of a feedforward network in the experiments.

On the other hand, the large number of input variables ($n_i = 90$, in the experiments) most likely explains the poor performance of the mixture of Gaussian model in comparison to both the static networks and the recurrent network. The Gaussian model requires estimating $O(n_i^2)$ parameters and inverting large covariance matrices.

The approach to handling missing values presented here can also be extended to sequential data with missing or asynchronous variables. As our experiments suggest, for such problems, using recurrence and multiple time scales yields better performance than static or time-delay networks for which the missing values are filled using a heuristic.

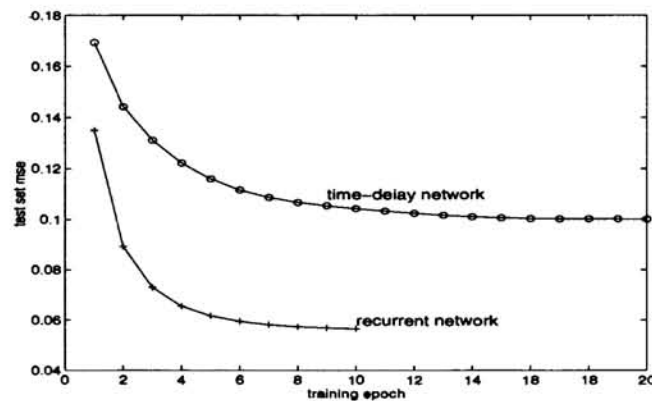

Figure 4: Test set mean squared error on the asynchronous data. Top: static network with time delays. Bottom: recurrent network with feedback to input values to complete missing data.

## References

Ahmad, S. and Tresp, V. (1993). Some solutions to the missing feature problem in vision. In Hanson, S. J., Cowan, J. D., and Giles, C. L., editors, *Advances in Neural Information Processing Systems 5*, San Mateo, CA. Morgan Kaufman Publishers.

Almeida, L. (1987). A learning rule for asynchronous perceptrons with feedback in a combinatorial environment. In Caudill, M. and Butler, C., editors, *IEEE International Conference on Neural Networks*, volume 2, pages 609–618, San Diego 1987. IEEE, New York.

Bareiss, E. and Porter, B. (1987). Protos: An exemplar-based learning apprentice. In *Proceedings of the 4th International Workshop on Machine Learning*, pages 12–23, Irvine, CA. Morgan Kaufmann.

Bottou, L. and Gallinari, P. (1991). A framework for the cooperation of learning algorithms. In Lippman, R. P., Moody, R., and Touretzky, D. S., editors, *Advances in Neural Information Processing Systems 3*, pages 781–788, Denver, CO.

Ghahramani, Z. and Jordan, M. I. (1994). Supervised learning from incomplete data via an EM approach. In Cowan, J., Tesauro, G., and Alspector, J., editors, *Advances in Neural Information Processing Systems 6*, page , San Mateo, CA. Morgan Kaufmann.

Lang, K. J., Waibel, A. H., and Hinton, G. E. (1990). A time-delay neural network architecture for isolated word recognition. *Neural Networks*, 3:23–43.

Pineda, F. (1989). Recurrent back-propagation and the dynamical approach to adaptive neural computation. *Neural Computation*, 1:161–172.

Simard, P. and LeCun, Y. (1992). Reverse TDNN: An architecture for trajectory generation. In Moody, J., Hanson, S., and Lipmann, R., editors, *Advances in Neural Information Processing Systems 4*, pages 579–588, Denver, CO. Morgan Kaufmann, San Mateo.

Tresp, V., Ahmad, S., and Neuneier, R. (1994). Training neural networks with deficient data. In Cowan, J., Tesauro, G., and Alspector, J., editors, *Advances in Neural Information Processing Systems 6*, pages 128–135. Morgan Kaufman Publishers, San Mateo, CA.
